# Searching for Character Models

**Jaety Edwards**
Department of Computer Science
UC Berkeley
Berkeley, CA 94720
jaety@cs.berkeley.edu

**David Forsyth**
Department of Computer Science
UC Berkeley
Berkeley, CA 94720
daf@cs.berkeley.edu

## Abstract

We introduce a method to automatically improve character models for a handwritten script without the use of transcriptions and using a minimum of document specific training data. We show that we can use searches for the words in a dictionary to identify portions of the document whose transcriptions are unambiguous. Using templates extracted from those regions, we retrain our character prediction model to drastically improve our search retrieval performance for words in the document.

## 1 Introduction

An active area of research in machine transcription of handwritten documents is reducing the amount and expense of supervised data required to train prediction models. Traditional OCR techniques require a large sample of hand segmented letter glyphs for training. This per character segmentation is expensive and often impractical to acquire, particularly if the corpora in question contain documents in many different scripts.

Numerous authors have presented methods for reducing the expense of training data by removing the need to segment individual characters. Both Kopec et al [3] and LeCun et al [5] have presented models that take as input images of lines of text with their ASCII transcriptions. Training with these datasets is made possible by explicitly modelling possible segmentations in addition to having a model for character templates.

In their research on "wordspotting", Lavrenko et al [4] demonstrate that images of entire words can be highly discriminative, even when the individual characters composing the word are locally ambiguous. This implies that images of many sufficiently long words should have unambiguous transcriptions, even when the character models are poorly tuned. In our previous work, [2], the discriminatory power of whole words allowed us to achieve strong search results with a model trained on a single example per character.

The above results have shown that A) one can learn new template models given images of text lines and their associated transcriptions, [3, 5] without needing an explicit segmentation and that B) entire words can often be identified unambiguously, even when the models for individual characters are poorly tuned. [2, 4]. The first of these two points implies that given a transcription, we can learn new character models. The second implies that for at least some parts of a document, we should be able to provide that transcription "for free", by matching against a dictionary of known words.

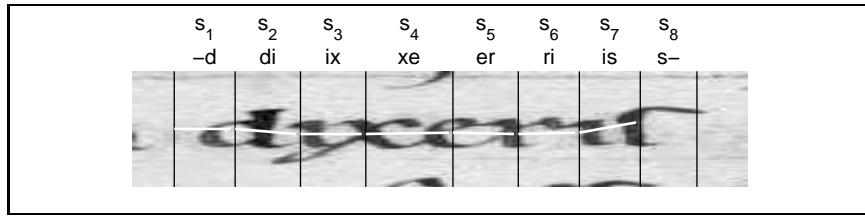

Figure 1: **A line, and the states that generate it.** *Each state $s_t$ is defined by its left and right characters $c_{tl}$ and $c_{tr}$ (eg "x" and "e" for $s_4$). In the image, a state spans half of each of these two characters, starting just past the center of the left character and extending to the center of the right character, i.e. the right half of the "x" and the left half of the "e" in $s_4$. The relative positions of the two characters is given by a displacement vector $d_t$ (superimposed on the image as white lines). Associating states with intracharacter spaces instead of with individual characters allows for the bounding boxes of characters to overlap while maintaining the independence properties of the Markov chain.*

In this work we combine these two observations in order to improve character models without the need for a document specific transcription. We provide a generic dictionary of words in the target language. We then identify "high confidence" regions of a document. These are image regions for which exactly one word from our dictionary scores highly under our model. Given a set of high confidence regions, we effectively have a training corpus of text images with associated transcriptions. In these regions, we infer a segmentation and extract new character examples. Finally, we use these new exemplars to learn an improved character prediction model. As in [2], our document in this work is a 12th century manuscript of Terence's Comedies obtained from Oxford's Bodleian library [1].

## 2 The Model

Hidden Markov Models are a natural and widely used method for modeling images of text. In their simplest incarnation, a hidden state represents a character and the evidence variable is some feature vector calculated at points along the line. If all characters were known to be of a single fixed width, this model would suffice. The probability of a line under this model is given as

$$p(line) = p(c_1|\alpha) \prod_{t>1} p(c_t|c_{t-1})p(im_{[w*(t-1):w*t]}|c_t) \tag{1}$$

where $c_t$ represents the $t^{th}$ character on the line, $\alpha$ represents the start state, $w$ is the width of a character, and $im_{[w(t-1)+1:wt]}$ represents the column of pixels beginning at column $w*(t-1)+1$ of the image and ending at column $w*t$, (i.e. the set of pixels spanned by $c$)

Unfortunately, character's widths do vary quite substantially and so we must extend the model to accommodate different possible segmentations. A generalized HMM allows us to do this. In this model a hidden state is allowed to emit a variable length series of evidence variables. We introduce an explicit distribution over the possible widths of a character. Letting $d_t$ be the displacement vector associated with the $t^{th}$ character, and $c_{tx}$ refer to the $x$ location of the left edge of a character on the line, the probability of a line under this revised model is

$$p(line) = p(c_1|\alpha) \prod_{t>1} p(c_t|c_{t-1})p(d_t|c_t)p(im_{[c_{tx}+1:c_{tx}+d]}|d_t, c_t) \tag{2}$$

This is the model we used in [2]. It performs far better than using an assumption of fixed widths, but it still imposes unrealistic constraints on the relative positions of characters. In

particular, the portion of the ink generated by the current character is assumed to be independent of the preceding character. In other words, the model assumes that the bounding boxes of characters do not overlap. This constraint is obviously unrealistic. Characters routinely overlap in our documents. "f"s, for instance, form ligatures with most following characters. In previous work, we treated this overlap as noise, hurting our ability to correctly localize templates. Under this model, local errors of alignment would also often propagate globally, adversely affecting the segmentation of the whole line. For search, this noisy segmentation still provides acceptable results. In this work, however, we need to extract new templates, and thus correct localization and segmentation of templates is crucial.

In our current work, we have relaxed this constraint, allowing characters to partially overlap. We achieve this by changing hidden states to represent character bigrams instead of single characters (Figure 1). In the image, a state now spans the pixels from just past the center of the left character to the pixel containing the center of the right character. We adjust our notation somewhat to reflect this change, letting $s_t$ now represent the $t^{th}$ hidden state and $c_{tl}$ and $c_{tr}$ be the left and right characters associated with $s$. $d_t$ is now the displacement vector between the centers of $c_{tl}$ and $c_{tr}$.

The probability of a line under this, our actual, model is

$$p(line) = p(s_1|\alpha) \prod_{t>1} p(s_t|s_{t-1})p(d_t|c_{tl}, c_{tr})p(im_{[s_{tx}+1:s_{tx}+d_t]}|c_{tl}, c_{tr}, d_t) \qquad (3)$$

This model allows overlap of bounding boxes, but it does still make the assumption that the bounding box of the current character does not extend past the center of the previous character. This assumption does not fully reflect reality either. In Figure 1, for example, the left descender of the x extends back further than the center of the preceding character. It does, however, accurately reflect the constraints within the heart of the line (excluding ascenders and descenders). In practice, it has proven to generate very accurate segmentations. Moreover, the errors we do encounter no longer tend to affect the entire line, since the model has more flexibility with which to readjust back to the correct segmentation.

## 2.1 Model Parameters

Our transition distribution between states is simply a 3-gram character model. We train this model using a collection of ASCII Latin documents collected from the web. This set does not include the transcriptions of our documents.

Conditioned on displacement vector, the emission model for generating an image chunk given a state is a mixture of gaussians. We associate with each character a set of image windows extracted from various locations in the document. We initialize these sets with one example a piece from our hand cut set (Figure 2). We adjust the probability of an image given the state to include the distribution over blocks by expanding the last term of Equation 3 to reflect this mixture. Letting $b_{ck}$ represent the $k^{th}$ exemplar in the set associated with character $c$, the conditional probability of an image region spanning the columns from $x$ to $x'$ is given as

$$p(im_{x:x'}|c_{tl}, c_{tr}, d_t) = \sum_{i,j} p(im_{x:x'}|b_{c_{tl}i}, b_{c_{tr}j}, d_t) \qquad (4)$$

In principle, the displacement vectors should now be associated with an individual block, not a character. This is especially true when we have both upper and lower case letters. However, our model does not seem particularly sensitive to this displacement distribution and so in practice, we have a single, fairly loose, displacement distribution per character.

Given a displacement vector, we can generate the maximum likelihood template image under our model by compositing the correct halves of the left and right blocks. Reshaping

the image window into a vector, the likelihood of an image window is then modeled as a gaussian, using the corresponding pixels in the template as the means, and assuming a diagonal covariance matrix. The covariance matrix largely serves to mask out empty regions of a character's bounding box, so that we do not pay a penalty when the overlap of two characters' bounding boxes contains only whitespace.

## 2.2 Efficiency Considerations

The number of possible different templates for a state is $O(|B| \times |B| \times |D|)$, where $|B|$ is the number of different possible blocks and $|D|$ is the number of candidate displacement vectors. To make inference in this model computationally feasible, we first restrict the domain of $d$. For a given pair of blocks $b_l$ and $b_r$, we consider only displacement vectors within some small $x$ distance from a mean displacement $m_{b_l,b_r}$, and we have a uniform distribution within this region. $m$ is initialized from the known size of our single hand cut template. In the current work, we do not relearn the $m$. These are held fixed and assumed to be the same for all blocks associated with the same letter.

Even when restricting the number of $d$'s under consideration as discussed above, it is computationally infeasible to consider every possible location and pair of blocks. We therefore prune our candidate locations by looking at the likelihood of blocks in isolation and only considering locations where there is a local optimum in the response function and whose value is better than a given threshold. In this case our threshold for a given location is that $\mathcal{L}(block) < .7\mathcal{L}(background)$ (where $\mathcal{L}(x)$ represents the negative log likelihood of $x$). In other words, a location has to look at least marginally more like a given block than it looks like the background.

After pruning locations in this manner, we are left with a discrete set of "sites," where we define a site as the tuple (block type, x location, y location). We can enumerate the set of possible states by looking at every pair of sites whose displacement vector has a non-zero probability.

## 2.3 Inference In The Model

The statespace defined above is a directed acyclic graph, anchored at the left edge and right edges of a line of text. A path through this lattice defines both a transcription and a segmentation of the line into individual characters. Inference in this model is relatively straightforward because of our constraint that each character may overlap only one preceding and one following character, and our restriction of displacement vectors to a small discrete range. The first restriction means that we need only consider binary relations between templates. The second preserves the independence relationships of an HMM. A given state $s_t$ is independent of the rest of the line given the values of all other states within $d_{max}$ of either edge of $s_t$ (where $d_{max}$ is the legal displacement vector with the longest $x$ component.) We can therefore easily calculate the best path or explicitly calculate the posterior of a node by traversing the state graph in topological order, sorted from left to right. The literature on Weighted Finite State Transducers ([6], [5]) is a good resource for efficient algorithms on these types of statespace graph.

# 3 Learning Better Character Templates

We initialize our algorithm with a set of handcut templates, exactly 1 per character, (Figure 2), and our goal is to construct more accurate character models automatically from unsupervised data. As noted above, we can easily calculate the posterior of a given site under our model. (Recall that a site is a particular character template at a given (x,y) location in the line.) The traditional EM approach to estimating new templates would be to use these

a b c d e f g h i l m n o p q r ſ t u u x y

Figure 2: **Original Training Data** *These 22 glyphs are our only document specific training data. We use the model based on these characters to extract the new examples shown below*

Aaaqaaauaa ſſſſſſ qqqqqqqqq
uoaaaaaa ſſſſſſ qqqqqqqq

Figure 3: **Examples of extracted templates** *We extract new templates from high confidence regions. From these, we choose a subset to incorporate into the model as new exemplars. Templates are chosen iteratively to best cover the space of training examples. Notice that for "q" and "a", we have extracted capital letters, of which there were no examples in our original set of glyphs. This happens when the combination of constraints from the dictionary the surrounding glyphs make a "q" or "a" the only possible explanation for this region, even though its local likelihood is poor.*

sites as training examples, weighted by their posteriors. Unfortunately, the constraints imposed by 3 and even 4-gram character models seem to be insufficient. The posteriors of sites are not discriminative enough to get learning off the ground.

The key to successfully learning new templates lies is the observation from our previous work [2], that even when the posteriors of individual characters are not discriminative, one can still achieve very good search results with the same model. The search word in effect serves as its own language model, only allowing paths through the state graph that actually contain it, and the longer the word the more it constrains the model. Whole words impose much tighter constraints than a 2 or 3-gram character model, and it is only with this added power that we can successfully learn new character templates.

We define the score for a search as the negative log likelihood of the best path containing that word. With sufficiently long words, it becomes increasingly unlikely that a spurious path will achieve a high score. Moreover, if we are given a large dictionary of words and no alternative word explains a region of ink nearly as well as the best scoring word, then we can be extremely confident that this is a true transcription of that piece of ink.

Starting with a weak character model, we do not expect to find many of these "high confidence" regions, but with a large enough document, we should expect to find some. From these regions, we can extract new, reliable templates with which to improve our character models. The most valuable of these new templates will be those that are significantly different from any in our current set. For example, in Figure 3, note that our system identifies capital Q's, even though our only input template was lower case. It identifies this ink as a Q in much the same way that a person solves a crossword puzzle. We can easily infer the missing character in the string "obv-ous" because the other letters constrain us to one possible solution. Similarly, if other character templates in a word match well, then we can unambiguously identify the other, more ambiguous ones. In our Latin case, "Quid" is the only likely explanation for "-uid".

### 3.1 Extracting New Templates and Updating The Model

Within a high confidence region we have both a transcription and a localization of template centers. It remains only to cut out new templates. We accomplish this by creating a template image for the column of pixels from the corresponding block templates and then assigning image pixels to the nearest template character (measured by Euclidean distance).

Given a set of templates extracted from high confidence regions, we choose a subset of

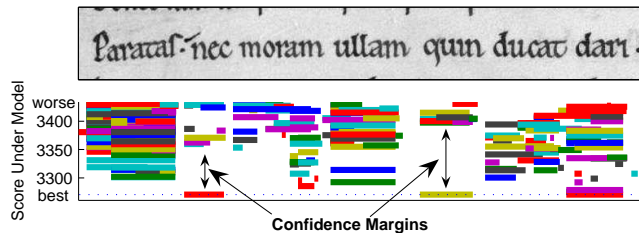

Figure 4: *Each line segment in the lower figure represents a proposed location for a word from our dictionary. It's vertical height is the score of that location under our model. A lower score represents a better fit. The dotted line is the score of our model's best possible path. Three correct words, "nec", "quin" and "dari", are actually on the best path. We define the* **confidence margin** *of a location as the difference in score between the best fitting word from our dictionary and the next best.*

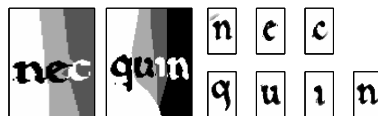

Figure 5: **Extracting Templates** *For a region with sufficiently high confidence margin, we construct the maximum likelihood template from our current exemplars.* **left***, and we assign pixels from the original image to a template based on its distance to the nearest pixel in the template image, extracting new glyph exemplars* **right***. These new glyphs become the exemplars for our next round of training.*

templates that best explain the remaining examples. We do this in a greedy fashion by choosing the example whose likelihood is lowest under our current model and adding it to our set. Currently, we threshold the number of new templates for the sake of efficiency. Finally, given the new set of templates, we can add them to the model and rerun our searches, potentially identifying new high confidence regions.

## 4   Results

Our algorithm iteratively improves the character model by gathering new training data from high confidence regions. Figure 3 shows that this method finds new templates significantly different from the originals. In this document, our set of examples after one round appears to cover the space of character images well, at least those in lower case. Our templates are not perfect. The "a", for instance, has become associated with at least one block that is in fact an "o". These mistakes are uncommon, particularly if we restrict ourselves to longer words. Those that do occur introduce a tolerable level noise into our model. They make certain regions of the document more ambiguous locally, but that local ambiguity can be overcome with the context provided by surrounding characters and a language model.

**Improved Character Models** We evaluate the method more quantitatively by testing the impact of the new templates on the quality of searches performed against the document. To search for a given word, we rank lines by the ratio of the maximum likelihood transcription/segmentation that contains the search word to the likelihood of the best possible segmentation/transcription under our model. The lowest possible search score is 1, happening when the search word is actually a substring of the maximum likelihood transcription. Higher scores mean that the word is increasingly unlikely under our model. In Figure 7, the figure on the left shows the improvement in ranking of the lines that truly contain selected search words. The odd rows (in red) are search results using only the original 22 glyphs,

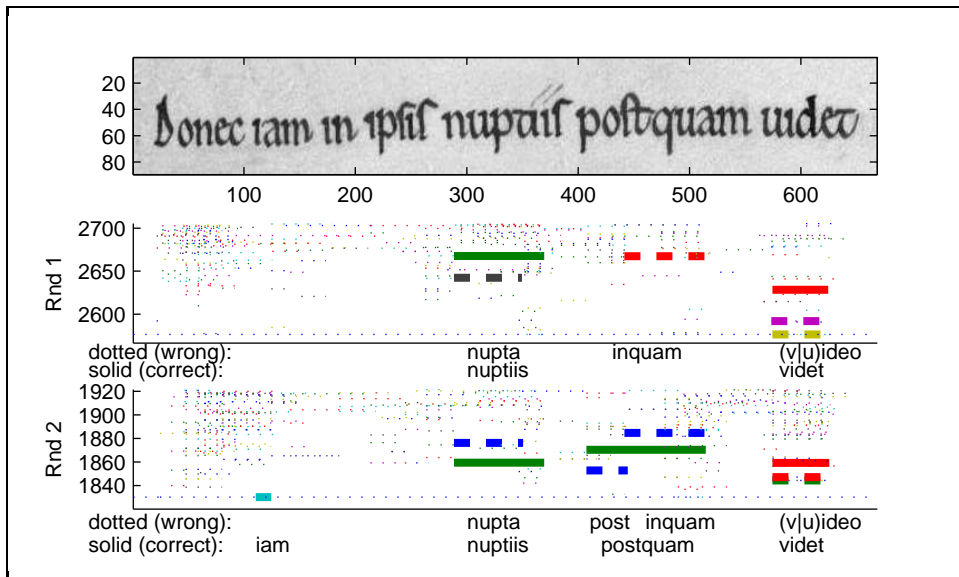

Figure 6: *Search Results with (Rnd 1) initial templates only and with (Rnd 2) templates extracted from high confidence regions. We show results that have a score within 5% of the best path.* **Solid Lines** *are the results for the correct word. Dotted lines represent other search results, where we have made a few larger in order to show those words that are the closest competitors to the true word. Many alternative searches, like the highlighted "post" are actually portions of the correct larger words. These restrict our selection of confidence regions, but do not impinge on search quality.*
*Each correct word has significantly improved after one round of template reestimation.* **"iam"** *has been correctly identified, and is a new high confidence region. Both* **"nuptiis"** *and* **"postquam"** *are now the highest likelihood words for their region barring smaller subsequences, and* **"videt"** *has narrowed the gap between its competitor "video".*

while the even rows (in green) use an additional 332 glyphs extracted from high confidence regions. Search results are markedly improved in the second model. The word "est", for instance, only had 15 of 24 of the correct lines in the top 100 under the original model, while under the learned model all 24 are not only present but also more highly ranked.

**Improved Search** *Figure 6* shows the improved performance of our refitted model for a single line. Most words have greatly improved relative to their next best alternative. "postquam" and "iam" were not even considered by the original model and now are nearly optimal. The *right of Figure 7* shows the average precision/recall curve under each model for 21 words with more than 4 occurrences in the dataset. Precision is the percentage of lines truly containing a word in the top $n$ search results, and recall is the percentage of all lines containing the word returned in the top $n$ results. The learned model clearly dominates. The new model also greatly improves performance for rare words. For 320 words ocurring just once in the dataset, 50% are correctly returned as the top ranked result under the original model. Under the learned model, this number jumps to 78%.

## 5   Conclusions and Future Work

In most fonts, characters are quite ambiguous locally. An "n" looks like a "u", looks like "ii", etc. This ambiguity is the major hurdle to the unsupervised learning of character templates. Language models help, but the standard n-gram models provide insufficient constraints, giving posteriors for character sites too uninformative to get EM off the ground.

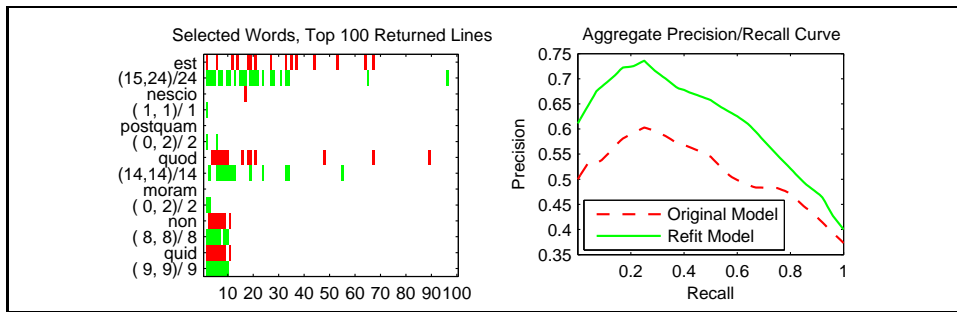

Figure 7: *The figure on the* **left** *shows the those lines with the top 100 scores that actually contain the specified word. The first of each set of two rows (in red) is the results from Round 1. The second (in green) is the results for Round 2. Almost all search words in our corpus show a significant improvement. The numbers to the right (x/y) mean that out of y lines that actually contained the search word in our document, x of them made it into the top ten. On the* **right** *are average precision/recall curves for 21 high frequency words under the model with our original templates (Rnd 1) and after refitting with new extracted templates (Rnd 2). Extracting new templates vastly improves our search quality*

An entire word is much different. Given a dictionary, we expect many word images to have a single likely transcription even if many characters are locally ambiguous. We show that we can identify these high confidence regions even with a poorly tuned character model. By extracting new templates only from these regions of the document, we overcome the noise problem and significantly improve our character models. We demonstrate this improvement for the task of search where the refitted models have drastically better search responses than with the original. Our method is indifferent to the form of the actual character emission model. There is a rich literature in character prediction from isolated image windows, and we expect that incorporating more powerful character models should provide even greater returns and help us in learning less regular scripts.

Finding high confidence regions to extract good training examples is a broadly applicable concept. We believe this work should extend to other problems, most notably speech recognition. Looked at more abstractly, our use of language model in this work is actually encoding spatial constraints. The probability of a character given an image window depends not only on the identify of surrounding characters but also on their spatial configuration. Integrating context into recognition problems is an area of intense research in the computer vision community, and we are investigating extending the idea of confidence regions to more general object recognition problems.

## References

[1] Early Manuscripts at Oxford University. Bodleian library ms. auct. f. 2.13. *http://image.ox.ac.uk/*.

[2] J. Edwards, Y.W. Teh, D. Forsyth, R. Bock, M. Maire, and G. Vesom. Making latin manuscripts searchable using ghmm's. In *NIPS 17*, pages 385–392. 2005.

[3] G. Kopec and M. Lomelin. Document-specific character template estimation. In *Proceedings, Document Image Recognition III, SPIE*, 1996.

[4] V. Lavrenko, T. Rath, and R. Manmatha. Holistic word recognition for handwritten historical documents. In *dial*, pages 278–287, 2004.

[5] Y. LeCun, L. Bottou, Y. Bengio, and P. Haffner. Gradient-based learning applied to document recognition. *Proceedings of the IEEE*, 86(11):2278–2324, 1998.

[6] M. Mohri, F. Pereira, and M. Riley. Weighted finite state transducers in speech recognition. *ISCA ITRW Automatic Speech Recognition*, pages 97–106, 2000.
